# Non-parametric Group Orthogonal Matching Pursuit for Sparse Learning with Multiple Kernels

**Vikas Sindhwani and Aurélie C. Lozano**
IBM T.J. Watson Research Center
Yorktown Heights, NY 10598
{vsindhw,aclozano}@us.ibm.com

## Abstract

We consider regularized risk minimization in a large dictionary of Reproducing kernel Hilbert Spaces (RKHSs) over which the target function has a sparse representation. This setting, commonly referred to as Sparse Multiple Kernel Learning (MKL), may be viewed as the non-parametric extension of group sparsity in linear models. While the two dominant algorithmic strands of sparse learning, namely convex relaxations using $l_1$ norm (e.g., Lasso) and greedy methods (e.g., OMP), have both been rigorously extended for group sparsity, the sparse MKL literature has so far mainly adopted the former with mild empirical success. In this paper, we close this gap by proposing a Group-OMP based framework for sparse MKL. Unlike $l_1$-MKL, our approach decouples the sparsity regularizer (via a direct $l_0$ constraint) from the smoothness regularizer (via RKHS norms), which leads to better empirical performance and a simpler optimization procedure that only requires a black-box single-kernel solver. The algorithmic development and empirical studies are complemented by theoretical analyses in terms of Rademacher generalization bounds and sparse recovery conditions analogous to those for OMP [27] and Group-OMP [16].

## 1  Introduction

Kernel methods are widely used to address a variety of learning problems including classification, regression, structured prediction, data fusion, clustering and dimensionality reduction [22, 23]. However, choosing an appropriate kernel and tuning the corresponding hyper-parameters can be highly challenging, especially when little is known about the task at hand. In addition, many modern problems involve multiple heterogeneous data sources (e.g. gene functional classification, prediction of protein-protein interactions) each necessitating the use of a different kernel. This strongly suggests avoiding the risks and limitations of single kernel selection by considering flexible combinations of multiple kernels. Furthermore, it is appealing to impose sparsity to discard noisy data sources. As several papers have provided evidence in favor of using multiple kernels (e.g. [19, 14, 7]), the multiple kernel learning problem (MKL) has generated a large body of recent work [13, 5, 24, 33], and become the focal point of the intersection between non-parametric function estimation and sparse learning methods traditionally explored in linear settings.

Given a convex loss function, the MKL problem is usually formulated as the minimization of empirical risk together with a mixed norm regularizer, e.g., the square of the sum of individual RKHS norms, or variants thereof, that have a close relationship to the Group Lasso criterion [30, 2]. Equivalently, this formulation may be viewed as simultaneous optimization of both the non-negative convex combination of kernels, as well as prediction functions induced by this combined kernel. In constraining the combination of kernels, the $l_1$ penalty is of particular interest as it encourages sparsity in the supporting kernels, which is highly desirable when the number of kernels considered is large. The MKL literature has rapidly evolved along two directions: one concerns scalability of op-

timization algorithms beyond the early pioneering proposals based on Semi-definite programming or Second-order Cone programming [13, 5] to simpler and more efficient alternating optimization schemes [20, 29, 24]; while the other concerns the use of $l_p$ norms [10, 29] to construct complex non-sparse kernel combinations with the goal of outperforming 1-norm MKL which, as reported in several papers, has demonstrated mild success in practical applications.

The class of Orthogonal Matching Pursuit techniques has recently received considerable attention, as a competitive alternative to Lasso. The basic OMP algorithm originates from the signal-processing community and is similar to forward greedy feature selection, except that it performs re-estimation of the model parameters in each iteration, which has been shown to contribute to improved accuracy. For linear models, some strong theoretical performance guarantees and empirical support have been provided for OMP [31] and its extension for variable group selection, Group-OMP [16]. In particular it was shown in [25, 9] that OMP and Lasso exhibit competitive theoretical performance guarantees. It is therefore desirable to investigate the use of Matching Pursuit techniques in the MKL framework and whether one may be able to improve upon existing MKL methods.

Our contributions in this paper are as follows. We propose a non-parametric kernel-based extension to Group-OMP [16]. In terms of the feature space (as opposed to function space) perspective of kernel methods, this allows Group-OMP to handle groups that can potentially contain infinite features. By adding regularization in Group-OMP, we allow it to handle settings where the sample size might be smaller than the number of features in any group. Rather than imposing a mixed $l_1$/RKHS-norm regularizer as in group-Lasso based MKL, a group-OMP based approach allows us to consider the exact sparse kernel selection problem via $l_0$ regularization instead. Note that in contrast to the group-lasso penalty, the $l_0$ penalty by itself has no effect on the smoothness of each individual component. This allows for a clear decoupling between the role of the smoothness regularizer (namely, an RKHS regularizer) and the sparsity regularizer (via the $l_0$ penalty). Our greedy algorithms allow for simple and flexible optimization schemes that only require a black-box solver for standard learning algorithms. In this paper, we focus on multiple kernel learning with Regularized least squares (RLS). We provide a bound on the Rademacher complexity of the hypothesis sets considered by our formulation. We derive conditions analogous to OMP [27] and Group-OMP [16] to guarantee the "correctness" of kernel selection. We close this paper with empirical studies on simulated and real-world datasets that confirm the value of our methods.

## 2 Learning Over an RKHS Dictionary

In this section, we setup some notation and give a brief background before introducing our main objective function and describing our algorithm in the next section. Let $\mathcal{H}_1 \ldots \mathcal{H}_N$ be a collection of Reproducing Kernel Hilbert Spaces with associated Kernel functions $k_1 \ldots k_N$ defined on the input space $\mathcal{X} \subset \mathbb{R}^d$. Let $\mathcal{H}$ denote the sum space of functions,

$$\mathcal{H} = \mathcal{H}_1 \oplus \mathcal{H}_2 \ldots \oplus \mathcal{H}_N = \{f : \mathcal{X} \mapsto \mathbb{R} | f(\mathbf{x}) = \sum_{j=1}^{N} f_j(\mathbf{x}), \mathbf{x} \in \mathcal{X}, f_j \in \mathcal{H}_j, j = 1 \ldots N\}$$

Let us equip this space with the following $l_p$ norms,

$$\|f\|_{l_p(\mathcal{H})} = \inf \left\{ \left( \sum_{j=1}^{N} \|f_j\|_{\mathcal{H}_j}^p \right)^{\frac{1}{p}} : f(\mathbf{x}) = \sum_{j=1}^{N} f_j(\mathbf{x}), \mathbf{x} \in \mathcal{X}, f_j \in \mathcal{H}_j, j = 1 \ldots N \right\} \quad (1)$$

It is now natural to consider a regularized risk minimization problem over such a RKHS dictionary, given a collection of training examples $\{\mathbf{x}_i, y_i\}_{i=1}^{l}$,

$$\underset{f \in \mathcal{H}}{\arg\min} \frac{1}{l} \sum_{i=1}^{l} V(y_i, f(\mathbf{x}_i)) + \lambda \|f\|_{l_p(\mathcal{H})}^2 \quad (2)$$

where $V(\cdot, \cdot)$ is a convex loss function such as squared loss in the Regularized Least Squares (RLS) algorithm or the hinge loss in the SVM method. If this problem again has elements of an RKHS structure, then, via the Representer Theorem, it can again be reduced to a finite dimensional problem and efficiently solved.

Let $q = \frac{p}{2-p}$ and let us define the $q$-convex hull of the set of kernel functions to be the following,

$$co_q(k_1 \ldots k_N) = \left\{ k_{\boldsymbol{\gamma}} : \mathcal{X} \times \mathcal{X} \mapsto \mathbb{R} \mid k_{\boldsymbol{\gamma}}(\mathbf{x}, \mathbf{z}) = \sum_{j=1}^{N} \gamma_j k_j(\mathbf{x}, \mathbf{z}), \sum_{j=1}^{N} \gamma_j^q = 1, \gamma_j \geq 0 \right\}$$

where $\boldsymbol{\gamma} \in \mathbb{R}^N$. It is easy to see that the non-negative combination of kernels, $k_{\boldsymbol{\gamma}}$, is itself a valid kernel with an associated RKHS $\mathcal{H}_{k_{\boldsymbol{\gamma}}}$. With this definition, [17] show the following,

$$\|f\|_{l_p(\mathcal{H})} = \inf_{\boldsymbol{\gamma}} \left\{ \|f\|_{\mathcal{H}_{k_{\boldsymbol{\gamma}}}}, k_{\boldsymbol{\gamma}} \in co_q(k_1 \ldots k_N) \right\} \tag{3}$$

This relationship connects Tikhonov regularization with $l_p$ norms over $\mathcal{H}$ to regularization over RKHSs parameterized by the kernel functions $k_{\boldsymbol{\gamma}}$. This leads to a large family of "multiple kernel learning" algorithms (whose variants are also sometimes referred to as $l_q$-MKL) where the basic idea is to solve an equivalent problem,

$$\underset{f \in \mathcal{H}_{k_{\boldsymbol{\gamma}}}, \boldsymbol{\gamma} \in \triangle^q}{\arg\min} \frac{1}{l} \sum_{i=1}^{l} V(y_i, f(\mathbf{x}_i)) + \lambda \|f\|_{\mathcal{H}_{k_{\boldsymbol{\gamma}}}}^2 \tag{4}$$

where $\triangle^q = \{\boldsymbol{\gamma} \in \mathbb{R}^N : \|\boldsymbol{\gamma}\|_q = 1, \forall_{j=1}^n \gamma_j \geq 0\}$. For a fixed $\boldsymbol{\gamma}$, the optimization over $f \in \mathcal{H}_{k_{\boldsymbol{\gamma}}}$ is recognizable as an RKHS problem for which a standard black box solver may be used. The weights $\boldsymbol{\gamma}$ may then optimized in an alternating minimization scheme, although several other optimization procedures are also be used (see e.g., [4]). The case where $p = 1$ is of particular interest in the setting when the size of the RKHS dictionary is large but the unknown target function can be approximated in a much smaller number of RKHSs. This leads to a large family of sparse multiple kernel learning algorithms that have a strong connection to the Group Lasso [2, 20, 29].

## 3   Multiple Kernel Learning with Group Orthogonal Matching Pursuit

Let us recall the $l_0$ pseudo-norm, which is the cardinality of the sparsest representation of $f$ in the dictionary, $\|f\|_{l_0(\mathcal{H})} = \min\{|J| : f = \sum_{j \in J} f_j\}$. We now pose the following exact sparse kernel selection problem,

$$\underset{f \in \mathcal{H}}{\arg\min} \frac{1}{l} \sum_{i=1}^{l} V(y_i, f(\mathbf{x}_i)) + \lambda \|f\|_{l_2(\mathcal{H})}^2 \quad \text{subject to} \quad \|f\|_{l_0(\mathcal{H})} \leq s \tag{5}$$

It is important to note the following: when using a dictionary of universal kernels, e.g., Gaussian kernels with different bandwidths, the presence of the regularization term $\|f\|_{l_2(\mathcal{H})}^2$ is critical (i.e., $\lambda > 0$) since otherwise the labeled data can be perfectly fit by any single kernel. In other words, the kernel selection problem is ill-posed. While conceptually simple, our formulation is quite different from those proposed earlier since the role of a smoothness regularizer (via the $\|f\|_{l_2(\mathcal{H})}^2$ penalty) is decoupled from the role of a sparsity regularizer (via the constraint on $\|f\|_{l_0(\mathcal{H})} \leq s$). Moreover, the latter is imposed directly as opposed through a $p = 1$ penalty making the spirit of our approach closer to Group Orthogonal Matching Pursuit (Group-OMP [16]) where groups are formed by very high-dimensional (infinite for Gaussian kernels) feature spaces associated with the kernels. It has been observed in recent work [10, 29] on $l_1$-MKL that sparsity alone does not lead it to improvements in real-world empirical tasks and hence several methods have been proposed to explore $l_q$-norm MKL with $q > 1$ in Eqn. 4, making MKL depart away from sparsity in kernel combinations. By contrast, we note that as $q \to \infty$, $p \to 2$. Our approach gives a direct knob *both* on smoothness (via $\lambda$) and sparsity (via $s$) with a solution path along these dimensions that differs from that offered by Group-Lasso based $l_q$-MKL as $q$ is varied. By combining $l_0$ pseudo-norm with RKHS norms, our method is conceptually reminiscent of the elastic net [32] (also see [26, 12, 21]). If kernels arise from different subsets of input variables, our approach is also related to sparse additive models [18].

Our algorithm, MKL-GOMP, is outlined below for regularized least squares. Extensions for other loss functions, e.g., hinge loss for SVMs, can also be similarly derived. In the description of the algorithm, our notation is as follows: For any function $f$ belonging to an RKHS $\mathcal{F}_k$ with kernel function $k(\cdot, \cdot)$, we denote the regularized objective function as, $\mathcal{R}_\lambda(f, \mathbf{y}) = \frac{1}{l} \sum_{i=1}^{l} (y_i - f(\mathbf{x}_i))^2 + \lambda \|f\|_{\mathcal{F}_k}$

where $\| \cdot \|_{\mathcal{F}}$ denotes the RKHS norm. Recall that the minimizer $f^{\star} = \arg\min_{f \in \mathcal{F}} \mathcal{R}_{\lambda}(f, \mathbf{y})$ is given by solving the linear system, $\boldsymbol{\alpha} = (\mathbf{K} + \lambda l I)^{-1} \mathbf{y}$ where $\mathbf{K}$ is the gram matrix of the kernel on the labeled data, and by setting $f^{\star}(\mathbf{x}) = \sum_{i=1}^{l} \alpha_i k(\mathbf{x}, \mathbf{x}_i)$. Moreover, the objective value achieved by the minimizer is: $\mathcal{R}_{\lambda}(f^{\star}, \mathbf{y}) = \lambda \mathbf{y}^T (\mathbf{K} + \lambda l \mathbf{I})^{-1} \mathbf{y}$. Note that MKL-GOMP should not be confused with Kernel Matching Pursuit [28] whose goal is different: it is designed to sparsify $\boldsymbol{\alpha}$ in a single-kernel setting. The MKL-GOMP procedure iteratively expands the hypothesis space, $\mathcal{H}_{\mathcal{G}^{(1)}} \subseteq \mathcal{H}_{\mathcal{G}^{(2)}} \ldots \subseteq \mathcal{H}_{\mathcal{G}^{(i)}}$, by greedily selecting kernels from a given dictionary, where $\mathcal{G}^{(i)} \subset \{1 \ldots N\}$ is a subset of indices and $\mathcal{H}_{\mathcal{G}} = \bigcup_{j \in \mathcal{G}} \mathcal{H}_j$. Note that each $\mathcal{H}_{\mathcal{G}}$ is an RKHS with kernel $\sum_{j \in \mathcal{G}} k_j$ (see Section 6 in [1]). The selection criteria is the best improvement, $I(f^{(i)}, \mathcal{H}_j)$, given by a new hypothesis space $\mathcal{H}_j$ in reducing the norm of the current residual $\mathbf{r}^{(i)} = \mathbf{y} - \boldsymbol{f}^{(i)}$ where $\boldsymbol{f}^{(i)} = [f^{(i)}(\mathbf{x}_1) \ldots f^{(i)}(\mathbf{x}_l)]^T$, by finding the best regularized (smooth) approximation. Note that since $\min_{g \in \mathcal{H}_j} \mathcal{R}_{\lambda}(g, \mathbf{r}) \leq \mathcal{R}_{\lambda}(0, \mathbf{r}) = \|\mathbf{r}\|^2$, the value of the improvement function,

$$I(f^{(i)}, \mathcal{H}_j) = \|\mathbf{r}^{(i)}\|_2^2 - \min_{g \in \mathcal{H}_j} \mathcal{R}_{\lambda}(g, \mathbf{r}^{(i)})$$

is always non-negative. Once a kernel is selected, the function is re-estimated by learning in $\mathcal{H}_{\mathcal{G}^{(i)}}$. Note that since $\mathcal{H}_{\mathcal{G}}$ is an RKHS whose kernel function is the sum $\sum_{j \in \mathcal{G}} k_j$, we can use a simple RLS linear system solver for refitting. Unlike group-Lasso based MKL, we do not need an iterative kernel reweighting step which essentially arises as a mechanism to transform the less convenient group sparsity norms into reweighted squared RKHS norms. MKL-GOMP converges when the best improvement is no better than $\epsilon$.

---

► *Input*: Data matrix $\mathbf{X} = [\mathbf{x}_1 \ldots \mathbf{x}_l]^T$, Label vector $\mathbf{y} \in \mathbb{R}^l$, Kernel Dictionary $\{k_j(\cdot, \cdot)\}_{j=1}^N$, Precision $\epsilon > 0$

► *Output*: Selected Kernels $\mathcal{G}^{(i)}$ and a function $f^{(i)} \in \mathcal{H}_{\mathcal{G}^{(i)}}$

► Initialization: $\mathcal{G}^{(0)} = \emptyset$, $f^{(0)} = 0$, set residual $\mathbf{r}^{(0)} = \mathbf{y}$

► **for** $i = 0, 1, 2, \ldots$

     1. *Kernel Selection*: For all $j \notin \mathcal{G}^{(i)}$, set:
$$I(f^{(i)}, \mathcal{H}_j) = \|\mathbf{r}^{(i)}\|_2^2 - \min_{g \in \mathcal{H}_j} \mathcal{R}_{\lambda}(g, \mathbf{r}^{(i)})$$
$$= \mathbf{r}^{(i)T} \left( \mathbf{I} - \lambda(\mathbf{K}_j + \lambda l \mathbf{I})^{-1} \right) \mathbf{r}^{(i)}$$
Pick $j^{(i)} = \arg\max_{j \notin \mathcal{G}^{(i)}} I(f^{(i)}, \mathcal{H}_j)$

     2. *Convergence Check*: **if** $\left( I(f^{(i)}, \mathcal{H}_{j^{(i)}}) \leq \epsilon \right)$ **break**

     3. *Refitting*: Set $\mathcal{G}^{(i+1)} = \mathcal{G}^{(i)} \bigcup \{j^{(i)}\}$. Set $f^{(i+1)}(\mathbf{x}) = \sum_{j=1}^l \alpha_j k(\mathbf{x}, \mathbf{x}_j)$
         where $k = \sum_{j \in \mathcal{G}^{(i+1)}} k_j$ and $\boldsymbol{\alpha} = \left( \sum_{j \in \mathcal{G}^{(i+1)}} \mathbf{K}_j + \lambda l \mathbf{I} \right)^{-1} \mathbf{y}$

     4. *Update Residual*: $\mathbf{r}^{(i+1)} = \mathbf{y} - \boldsymbol{f}^{(i+1)}$ where $\boldsymbol{f}^{(i+1)} = [f^{(i+1)}(\mathbf{x}_1) \ldots f^{(i+1)}(\mathbf{x}_l)]^T$.

**end**

---

**Remarks:** Note that our algorithm can be applied to multivariate problems with group structure among outputs similar to Multivariate Group-OMP [15]. In particular, in our experiments on multiclass datasets, we treat all outputs as a single group and evaluate each kernel for selection based on how well the total residual is reduced across all outputs simultaneously. Kernel matrices are normalized to unit trace or to have uniform variance of data points in their associated feature spaces, as in [10, 33]. In practice, we can also monitor error on a validation set to decide the optimal degree of sparsity. For efficiency, we can precompute the matrices $\mathbf{Q}_j = (\mathbf{I} - \lambda(\mathbf{K}_j + \lambda l \mathbf{I})^{-1})^{\frac{1}{2}}$ so that $I(f^{(i)}, \mathcal{H}_j) = \|\mathbf{Q}_j \mathbf{r}\|_2^2$ can be very quickly evaluated at selection time, and/or reduce the search space by considering a random subsample of the dictionary.

## 4 Theoretical Analysis

Our analysis is composed of two parts. In the first part, we establish generalization bounds for the hypothesis spaces considered by our formulation, based on the notion of Rademacher complex-

ity. The second component of our theoretical analysis consists of deriving conditions under which MKL-GOMP can recover good solutions. While the first part can be seen as characterizing the "statistical convergence" of our method, the second part characterizes its "numerical convergence" as an optimization method, and is required to complement the first part. This is because matching pursuit methods can be deemed to solve an exact sparse problem approximately, while regularized methods (e.g. $l_1$ norm MKL) solve an approximate problem exactly. We therefore need to show that MKL-GOMP recovers a solution that is close to an optimum solution of the exact sparse problem.

## 4.1 Rademacher Bounds

**Theorem 1.** *Consider the hypothesis space of sufficiently sparse and smooth functions[1],*

$$\mathcal{H}_{\tau,s} = \left\{ f \in \mathcal{H} : \|f\|_{l_2(\mathcal{H})}^2 \leq \tau, \|f\|_{l_0(\mathcal{H})} \leq s \right\}$$

*Let $\delta \in (0,1)$ and $\kappa = \sup_{\mathbf{x} \in \mathcal{X}, j=1...N} k_j(\mathbf{x}, \mathbf{x})$. Let $\rho$ be any probability distribution on $(\mathbf{x}, y) \in \mathbf{X} \times \mathbb{R}$ satisfying $|y| \leq M$ almost surely, and let $\{\mathbf{x}_i, y_i\}_{i=1}^l$ be randomly sampled according to $\rho$. Define, $\hat{f} = \arg\min_{f \in \mathcal{H}_{\tau,s}} \frac{1}{l} \sum_{i=1}^l (y_i - f(\mathbf{x}_i))^2$ to be the empirical risk minimizer and $f^\star = \arg\min_{f \in \mathcal{H}_{\tau,s}} R(f)$ to be the true risk minimizer in $\mathcal{H}_{\tau,s}$ where $R(f) = \mathbb{E}_{(\mathbf{x},y)\sim\rho} (y - f(x))^2$ denotes the true risk. Then, with probability atleast $1 - \delta$ over random draws of samples of size l,*

$$R(\hat{f}) \leq R(f^\star) + 8L\sqrt{\frac{s\kappa\tau}{l}} + 4L^2\sqrt{\frac{\log(\frac{3}{\delta})}{2l}} \tag{6}$$

*where $\|y - f\|_\infty \leq L = (M + \sqrt{s\kappa\tau})$.*

The proof is given in supplementary material, but can also be reasoned as follows. In the standard single-RKHS case, the Rademacher complexity can be upper bounded by a quantity that is proportional to the square root of the trace of the Gram matrix, which is further upper bounded by $\sqrt{l\kappa}$. In our case, any collection of $s$-sparse functions from a dictionary of $N$ RKHSs reduces to a single RKHS whose kernel is the sum of $s$ base kernels, and hence the corresponding trace can be bounded by $\sqrt{ls\kappa}$ for all possible subsets of size $s$. Once it is established that the empirical Rademacher complexity of $\mathcal{H}_{\lambda,s}$ is upper bounded by $\sqrt{\frac{s\kappa\tau}{l}}$, the generalization bound follows from well-known results [6] tailored to regularized least squares regression with bounded target variable.

For $l_1$-norm MKL, in the context of margin-based loss functions, Cortes et. al., 2010 [8] bound the Rademacher complexity as $\sqrt{\frac{ce\lceil log(N)\rceil\kappa\tau}{l}}$ where $\lceil\cdot\rceil$ is the ceiling function that rounds to next integer, $e$ is the exponential and $c = \frac{23}{22}$. Using VC-based lower-bound arguments, they point out that the $\sqrt{log(N)}$ dependence on $N$ is essentially optimal. By contrast, our greedy approach with sequential regularized risk minimization imposes direct control over degree of sparsity as well as smoothness, and hence the Rademacher complexity in our case is independent of $N$. If $s = O(logN)$, the bounds are similar. A critical difference between $l_1$-norm MKL and sparse greedy approximations, however, is that the former is convex and hence the empirical risk can be minimized exactly in the hypothesis space whose complexity is bounded by Rademacher analysis. This is not true in our case, and therefore, to complement Rademacher analysis, we need conditions under which good solutions can be recovered.

## 4.2 Exact Recovery Conditions in Noiseless Settings

We now assume that the regression function $f_\rho(x) = \int y d\rho(y|x)$ is sparse, i.e., $f_\rho \in \mathcal{H}_{\mathcal{G}_{good}}$ for some subset $\mathcal{G}_{good}$ of $s$ "good" kernels and that it is sufficiently smooth in the sense that for some $\lambda > 0$, given sufficient samples, the empirical minimizer $\hat{f} = \arg\min_{f \in \mathcal{H}_{\mathcal{G}_{good}}} \mathcal{R}_\lambda(f, \mathbf{y})$ gives near optimal generalization as per Theorem 1. In this section our main concern is to characterize Group-OMP like conditions under which MKL-GOMP will be able to learn $\hat{f}$ by recovering the support $\mathcal{G}_{good}$ exactly.

Let us denote $r^{(i)} = \hat{f} - f^{(i)}$ as the *residual function* at step $i$ of the algorithm. Initially, $r^{(0)} = \hat{f} \in \mathcal{H}_{\mathcal{G}_{good}}$. Our argument is inductive: if at any step $i$, $r^{(i)} \in \mathcal{H}_{\mathcal{G}_{good}}$ *and* we can always guarantee that $\max_{j \in \mathcal{G}_{good}} I(f^{(i)}, \mathcal{H}_j) > \max_{j \notin \mathcal{G}_{good}} I(f^{(i)}, \mathcal{H}_j)$, i.e., a good kernel offers better greedy improvement, then it is clear that the algorithm correctly expands the hypothesis space and never makes a mistake. Without loss of generality, let us rearrange the dictionary so that $\mathcal{G}_{good} = \{1 \dots s\}$. For any function $f \in \mathcal{H}_{\mathcal{G}_{good}}$, we now wish to derive the following upper bound,

$$\frac{\|(I(f, \mathcal{H}_{s+1}) \dots I(f, \mathcal{H}_N))\|_\infty}{\|(I(f, \mathcal{H}_1) \dots I(f, \mathcal{H}_s))\|_\infty} \le \mu_{\mathcal{H}}(\mathcal{G}_{good})^2 \tag{7}$$

Clearly, a sufficient condition for exact recovery is $\mu_{\mathcal{H}}(\mathcal{G}_{good}) < 1$.

We need some notation to state our main result. Let $s = |\mathcal{G}_{good}|$, i.e., the number of good kernels. For any matrix $\mathbf{A} \in \mathbb{R}^{ls \times l(N-s)}$, let $\|\mathbf{A}\|_{(2,1)}$ denote the matrix norm induced by the following vector norms: for any vector $\mathbf{u} = [\mathbf{u}_1 \dots \mathbf{u}_s] \in \mathbb{R}^{ls}$ define $\|\mathbf{u}\|_{(2,1)} = \sum_{i=1}^s \|\mathbf{u}_i\|_2$; and similarly, for any vector $\boldsymbol{v} = [\boldsymbol{v}_1 \dots \boldsymbol{v}_{N-s}] \in \mathbb{R}^{l(N-s)}$ define $\|\boldsymbol{v}\|_{(2,1)} = \sum_{i=1}^{N-s} \|\boldsymbol{v}_i\|_2$. Then, $\|\mathbf{A}\|_{(2,1)} = \sup_{\boldsymbol{v} \in \mathbb{R}^{l(N-s)}} \frac{\|\mathbf{A}\boldsymbol{v}\|_{(2,1)}}{\|\boldsymbol{v}\|_{(2,1)}}$. We can now state the following:

**Theorem 2.** *Given the kernel dictionary $\{k_j(\cdot, \cdot)\}_{j=1}^N$ with associated gram matrices $\{\mathbf{K}_j\}_{i=1}^N$ over the labeled data, MKL-GOMP correctly recovers the good kernels, i.e., $\mathcal{G}^{(s)} = \mathcal{G}_{good}$, if*

$$\mu_{\mathcal{H}}(\mathcal{G}_{good}) = \|\mathbf{C}_{\lambda, \mathcal{H}}(\mathcal{G}_{good})\|_{(2,1)} < 1$$

*where $\mathbf{C}_{\lambda, \mathcal{H}}(\mathcal{G}_{good}) \in \mathbb{R}^{ls \times l(N-s)}$ is a coherence matrix whose $(i,j)^{th}$ block of size $l \times l$, $i \in \mathcal{G}_{good}, j \notin \mathcal{G}_{good}$, is given by,*

$$\mathbf{C}_{\lambda, \mathcal{H}}(\mathcal{G}_{good})_{i,j} = \mathbf{K}_{\mathcal{G}_{good}} \mathbf{Q}_i \left( \sum_{k \in \mathcal{G}_{good}} \mathbf{Q}_k \mathbf{K}_{\mathcal{G}_{good}}^2 \mathbf{Q}_k \right)^{-1} \mathbf{Q}_j \mathbf{K}_{\mathcal{G}_{good}} \tag{8}$$

*where $\mathbf{K}_{\mathcal{G}_{good}} = \sum_{j \in \mathcal{G}_{good}} \mathbf{K}_j$, $\mathbf{Q}_j = (\mathbf{I} - \lambda(\mathbf{K}_j + \lambda l \mathbf{I})^{-1})^{\frac{1}{2}}, j = 1 \dots N$.*

The proof is given in supplementary material. This result is analogous to sparse recovery conditions for OMP and $l_1$ methods and their (linear) group counterparts. In the noiseless setting, Tropp [27] gives an exact recovery condition of the form $\|\mathbf{X}_{good}^\dagger \mathbf{X}_{bad}\|_1 < 1$, where $\mathbf{X}_{good}$ and $\mathbf{X}_{bad}$ refer to the restriction of the data matrix to good and bad features, and $\|\cdot\|_1$ refers to the $l_1$ induced matrix norm. Intriguingly, the same paper shows that this condition is also sufficient for the Basis Pursuit $l_1$ minimization problem. For Group-OMP [16], the condition generalizes to involve a group sensitive matrix norm on the same matrix objects. Likewise, Bach [2] generalizes the Lasso variable selection consistency conditions to apply to Group Lasso and then further to non-parametric $l_1$-MKL. The above result is similar in spirit. A stronger sufficient condition can be derived by requiring $\|\mathbf{Q}_j \mathbf{K}_{\mathcal{G}_{good}}\|_2$ to be sufficiently small for all $j \notin \mathcal{G}_{good}$. Intuitively, this means that smooth functions in $\mathcal{H}_{\mathcal{G}_{good}}$ cannot be well approximated by using smooth functions induced by the "bad" kernels, so that MKL-GOMP is never led to making a mistake.

## 5  Empirical Studies

We report empirical results on a collection of simulated datasets and 3 classification problems from computational cell biology. In all experiments, as in [10, 33], candidate kernels are normalized multiplicatively to have uniform variance of data points in their associated feature spaces.

### 5.1  Adaptability to Data Sparsity - Simulated Setting

We adapt the experimental setting proposed by [10] where the sparsity of the target function is explicitly controlled, and the optimal subset of kernels is varied from requiring the entire dictionary to requiring a single kernel. Our goal is to study the solution paths offered by MKL-GOMP in comparison to $l_q$-norm MKL. For consistency, we use squared loss in all experiments[2]. We implemented

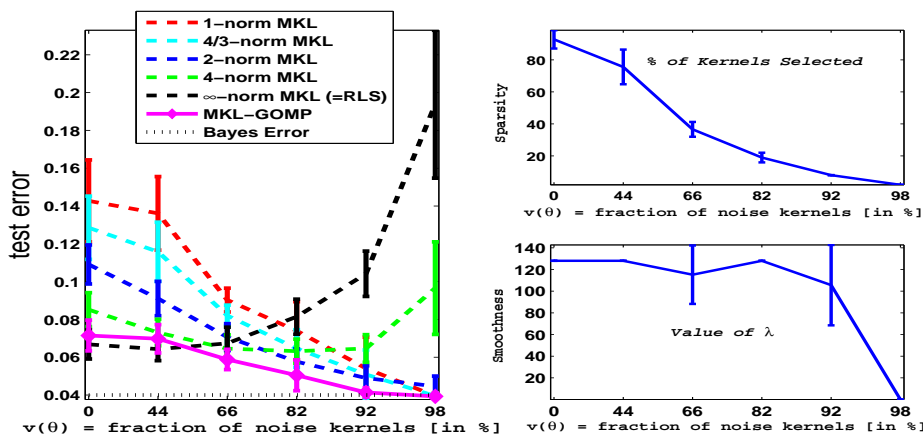

Figure 1: Simulated Setting: Adaptability to Data Sparsity

$l_q$-norm MKL for regularized least squares (RLS) using an alternating minimization scheme adapted from [17, 29]. Different binary classification datasets[3] with 50 labeled examples are randomly generated by sampling the two classes from 50-dimensional isotropic Gaussian distributions with equal covariance matrices (identity) and equal but opposite, means $\mu_1 = 1.75\frac{\theta}{\|\theta\|}$ and $\mu_2 = -\mu_1$ where $\theta$ is a binary vector encoding the true underlying sparsity. The fraction of zero components in $\theta$ is a measure for the feature sparsity of the learning problem. For each dataset, a linear kernel (normalized as in [10]) is generated from each feature and the resulting dictionary is input to MKL-GOMP and $l_q$-norm MKL. For each level of sparsity, a training of size 50, validation and test sets of size 10000 are generated 10 times and average classification errors are reported. For each run, the validation error is monitored as kernel selection progresses in MKL-GOMP and the number of kernels with smallest validation error are chosen. The regularization parameters for both MKL-GOMP and $l_q$ norm MKL are similarly chosen using the validation set. Figure 5.1 shows test error rates as a function of sparsity of the target function: from non-sparse (all kernels needed) to extremely sparse (only 1 kernel needed). We recover the observations also made in [10]: $l_1$-norm MKL excels in extremely sparse settings where a single kernel carries the whole discriminative information of the learning problem. However, in the other scenarios it mostly performs worse than the other $q > 1$ variants, despite the fact that the vector $\theta$ remains sparse in all but the uniform scenario. As $q$ is increased, the error rate in these settings improves but deteriorates in sparse settings. As reported in [11], the elastic net MKL approach of [26] performs similar to $l_1$-MKL in the hinge loss case. As can be seen in the Figure, the error curve of MKL-GOMP tends to be below the lower envelope of the error rates given by $l_q$-MKL solutions. To adapt to the sparsity of the problem, $l_q$ methods clearly need to tune $q$ requiring several fresh invocations of the appropriate $l_q$-MKL solver. On the other hand, in MKL-GOMP the hypothesis space grows as function of the iteration number and the solution trajectory naturally expands sequentially in the direction of decreasing sparsity. The right plot in Figure 5.1 shows the number of kernels selected by MKL-GOMP and the optimal value of $\lambda$, suggesting that MKL-GOMP adapts to the sparsity and smoothness of the learning problem.

## 5.2 Protein Subcellular Localization

The multiclass generalization of $l_1$-MKL proposed in [33] (MCMKL) is state of the art methodology in predicting protein subcellular localization, an important cell biology problem that concerns the estimation of where a protein resides in a cell so that, for example, the identification of drug targets can be aided. We use three multiclass datasets: PSORT+, PSORT- and PLANT provided by the authors of [33] at `http://www.fml.tuebingen.mpg.de/raetsch/suppl/protsubloc` together with a dictionary of 69 kernels derived with biological insight: 2 kernels on phylogenetic trees, 3 kernels based on similarity to known proteins (BLAST E-values), and 64 kernels based on amino-acid sequence patterns. The statistics of the three datasets are as follows: PSORT+ has 541 proteins labeled with 4 location classes, PSORT- has 1444 proteins in 5 classes and PLANT is

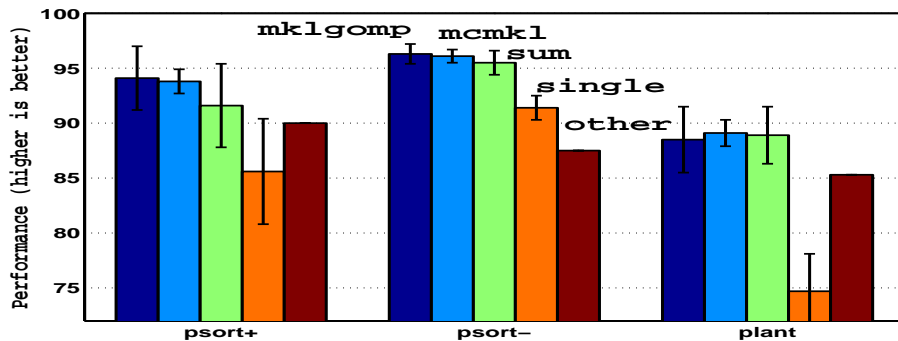

Figure 2: Protein Subcellular Localization Results

a 4-class problem with 940 proteins. For each dataset, results are averaged over 10 splits of the dataset into training and test sets. We used exactly the same experimental protocol, data splits and evaluation methodology as given in [33]: the hyper-parameters of MKL-GOMP (sparsity and the regularization parameter $\lambda$) were tuned based on 3-fold cross-validation; results on PSORT+, PSORT-are F1-scores averaged over the classes while those on PLANT are Mathew's correlation coefficient[4]. Figure 2 compare MKL-GOMP against MCMKL, baselines such as using the sum of all the kernels and using the best single kernel, and results from other prediction systems proposed in the literature. As can be seen, MKL-GOMP slightly outperforms MCMKL on PSORT+ an PSORT- datasets and is slightly worse on PLANT where RLS with the sum of all the kernels also performs very well. On the two PSORT datasets, [33] report selecting 25 kernels using MCMKL. On the other hand, on average, MKL-GOMP selects 14 kernels on PSORT+, 15 on PSORT- and 24 kernels on PLANT. Note that MKL-GOMP is applied in multivariate mode: the kernels are selected based on their utility to reduce the total residual error across all target classes.

## 6 Conclusion

By proposing a Group-OMP based framework for sparse multiple kernel learning, analyzing theoretically the performance of the resulting methods in relation to the dominant convex relaxation-based approach, and demonstrating the value of our framework through extensive experimental studies, we believe greedy methods arise as a natural alternative for tackling MKL problems. Relevant directions for future research include extending our theoretical analysis to the stochastic setting, investigating complex multivariate structures and groupings over outputs, e.g., by generalizing the multivariate version of Group-OMP [15], and extending our algorithm to incorporate interesting structured kernel dictionaries [3].

**Acknowledgments**

We thank Rick Lawrence, David S. Rosenberg and Ha Quang Minh for helpful conversations and support for this work.

## Footnotes

[1]Note that Tikhonov regularization using a penalty term $\lambda\|\cdot\|^2$, and Ivanov Regularization which uses a ball constraint $\|\cdot\|^2 \leq \tau$ return identical solutions for some one-to-one correspondence between $\lambda$ and $\tau$.

[2] $l_q$-MKL with SVM hinge loss behaves similarly.

[3]Provided by the authors of [10] at `mldata.org/repository/data/viewslug/mkl-toy/`

## References

[1] N. Aronszajn. Theory of reproducing kernel hilbert spaces. *Transactions of the American Mathematical Society*, 68(3):337–404, 1950.

[2] F. Bach. Consistency of group lasso and multiple kernel learning. *JMLR*, 9:1179–1225, 2008.

[3] F. Bach. High-dimensional non-linear variable selection through hierarchical kernel learning. In *Technical report, HAL 00413473*, 2009.

[4] F. Bach, R. Jenatton, J. Mairal, and G. Obozinski. Optimization with sparsity-inducing penalties. In *Technical report, HAL 00413473*, 2010.

---

[4] see http://www.fml.tuebingen.mpg.de/raetsch/suppl/protsubloc/protsubloc-wabi08-supp.pdf

[5] F. R. Bach, G. R. G. Lanckriet, and M. I. Jordan. Multiple kernel learning, conic duality, and the smo algorithm. In *ICML*, 2004.

[6] P. Bartlett and S. Mendelson. Rademacher and gaussian complexities: Risk bounds and structural results. *JMLR*, 3:463–482, 2002.

[7] A. Ben-Hur and W. S. Noble. Kernel methods for predicting protein–protein interactions. *Bioinformatics*, 21, January 2005.

[8] C. Cortes, M. Mohri, and Afshin Rostamizadeh. Generalization bounds for learning kernels. In *ICML*, 2010.

[9] A. K. Fletcher and S. Rangan. Orthogonal matching pursuit from noisy measurements: A new analysis. In *NIPS*, 2009.

[10] M. Kloft, U. Brefeld, S. Sonnenburg, and A. Zien. $l_p$-norm multiple kernel learning. *JMLR*, 12:953–997, 2011.

[11] M. Kloft, U. Ruckert, and P. Bartlett. A unifying view of multiple kernel learning. In *European Conference on Machine Learning (ECML)*, 2010.

[12] V. Koltchinskii and M. Yuan. Sparsity in multiple kernel learning. *The Annals of Statistics*, 38(6):3660–3695, 2010.

[13] G. R. G. Lanckriet, N. Cristianini, P. Bartlett, L. El Ghaoui, and M. I. Jordan. Learning the kernel matrix with semidefinite programming. *J. Mach. Learn. Res.*, 5:27–72, December 2004.

[14] G. R. G. Lanckriet, T. De Bie, N. Cristianini, M. I. Jordan, and W. S. Noble. A statistical framework for genomic data fusion. *Bioinformatics*, 20, November 2004.

[15] A. C. Lozano and V. Sindhwani. Block variable selection in multivariate regression and high-dimensional causal inference. In *NIPS*, 2010.

[16] A. C. Lozano, G. Swirszcz, and N. Abe. Group orthogonal matching pursuit for variable selection and prediction. In *NIPS*, 2009.

[17] C. Michelli and M. Pontil. Learning the kernel function via regularization. *JMLR*, 6:1099–1125, 2005.

[18] H. Liu P. Ravikumar, J. Lafferty and L. Wasserman. Sparse additive models. *Journal of the Royal Statistical Society: Series B (Statistical Methodology) (JRSSB)*, 71 (5):1009–1030, 2009.

[19] P. Pavlidis, J. Cai, J. Weston, and W.S. Noble. Learning gene functional classifications from multiple data types. *Journal of Computational Biology*, 9:401–411, 2002.

[20] A. Rakotomamonjy, F.Bach, S. Cano, and Y. Grandvalet. SimpleMKL. *Journal of Machine Learning Research*, 9:2491–2521, 2008.

[21] G. Raskutti, M. Wainwrigt, and B. Yu. Minimax-optimal rates for sparse additive models over kernel classes via convex programming. In *Technical Report 795, Statistics Department, UC Berkeley.*, 2010.

[22] Bernhard Scholkopf and Alexander J. Smola. *Learning with Kernels: Support Vector Machines, Regularization, Optimization, and Beyond*. MIT Press, 2001.

[23] J. Shawe-Taylor and N. Cristianini. *Kernel Methods for Pattern Analysis*. Cambridge University Press, 2004.

[24] S. Sonnenburg, G. Rätsch, C. Schäfer, and B. Schölkopf. Large scale multiple kernel learning. *J. Mach. Learn. Res.*, 7, December 2006.

[25] Zhang T. Sparse recovery with orthogonal matching pursuit under rip. *Computing Research Repository*, 2010.

[26] R. Tomioka and T. Suzuki. Sparsity-accuracy tradeoff in mkl. In *NIPS Workshop: Understanding Multiple Kernel Learning Methods. Technical report, arXiv:1001.2615v1*, 2010.

[27] J. Tropp. Greed is good: Algorithmic results for sparse approximation. *IEEE Trans. Inform. Theory,*, 50(10):2231–2242, 2004.

[28] P. Vincent and Y. Bengio. Kernel matching pursuit. *Machine Learning*, 48:165–188, 2002.

[29] Z. Xu, R. Jin, H. Yang, I. King, and M.R. Lyu. Simple and efficient multiple kernel learning by group lasso. In *ICML*, 2010.

[30] Ming Yuan, Ali Ekici, Zhaosong Lu, and Renato Monteiro. Dimension reduction and coefficient estimation in multivariate linear regression. *Journal Of The Royal Statistical Society Series B*, 69(3):329–346, 2007.

[31] Tong Zhang. On the consistency of feature selection using greedy least squares regression. *J. Mach. Learn. Res.*, 10, June 2009.

[32] H. Zhou and T. Hastie. Regularization and variable selection via the elastic net. *Journal of the Royal Statistical Society*, 67(2):301–320, 2005.

[33] A. Zien and Cheng S. Ong. Multiclass multiple kernel learning. ICML, 2007.

